# Risk Sensitive Particle Filters

**Sebastian Thrun, John Langford, Vandi Verma**
School of Computer Science
Carnegie Mellon University
Pittsburgh, PA 15213
*{thrun,jcl,vandi}@cs.cmu.edu*

## Abstract

We propose a new particle filter that incorporates a model of costs when generating particles. The approach is motivated by the observation that the costs of accidentally not tracking hypotheses might be significant in some areas of state space, and next to irrelevant in others. By incorporating a cost model into particle filtering, states that are more critical to the system performance are more likely to be tracked. Automatic calculation of the cost model is implemented using an MDP value function calculation that estimates the value of tracking a particular state. Experiments in two mobile robot domains illustrate the appropriateness of the approach.

## 1 Introduction

In recent years, particle filters [3, 7, 8] have found widespread application in domains with noisy sensors, such as computer vision and robotics [2, 5]. Particle filters are powerful tools for Bayesian state estimation in non-linear systems. The key idea of particle filters is to approximate a posterior distribution over unknown state variables by a set of particles, drawn from this distribution.

This paper addresses a primary deficiency of particle filters: Particle filters are insensitive to costs that might arise from the approximate nature of the particle representation. Their only criterion for generating a particle is the posterior likelihood of a state. To illustrate this point, consider the example of a Space Shuttle. Failures of the engine system are extremely unlikely, even in the presence of evidence to the contrary. Should we therefore not track the possibility of such failures, just because they are unlikely? If failure to track such low-likelihood events may incur high costs—such as a mission failure—these variables should be tracked even when their posterior probability is low. This observation suggests that costs should be taken into consideration when generating particles in the filtering process.

This paper proposes a particle filter that generates particles according to a distribution that combines the posterior probability with a risk function. The risk function measures the importance of a state location on future cumulative costs. We obtain this risk function via an MDP that calculates the approximate future risk of decisions made in a particular state. Experimental results in two robotic domains illustrate that our approach yields significantly better results than a particle filter insensitive to costs.

## 2  The "Classical" Particle Filter

Particle filters are a popular means of estimating the state of partially observable controllable Markov chains [3], sometimes referred to as dynamical systems [1]. To do so, particle filters require two types of information: *data*, and *a probabilistic generative model of the system*. The data generally comes in two flavors: measurements (e.g., camera images) and controls (e.g., robot motion commands). The measurement at time $t$ will be denoted $z_t$, and $u_t$ denotes the control asserted in the time interval $(t-1, t]$. Thus, the data is given by

$$z^t = z_1, z_2, \ldots, z_t \qquad \text{and} \qquad u^t = u_1, u_2, \ldots, u_t$$

Following common notation in the controls literature, we use the subscript $_t$ to refer to an event at time $t$ and the superscript $^t$ to denote all events leading up to time $t$.

Particle filters, like any member of the family of Bayes filters such as Kalman filters and HMMs, estimate the posterior distribution of the state of the dynamical system conditioned on the data, $p(x_t|z^t, u^t)$. They do so via the following recursive formula

$$p(x_t|z^t, u^t) = \eta_t \, p(z_t|x_t) \int p(x_t|u_t, x_{t-1}) \, p(x_{t-1}|z^{t-1}, u^{t-1}) \, dx_{t-1} \qquad (1)$$

where $\eta_t$ is a normalization constant. To calculate this posterior, three probability distributions are required, which together are commonly referred as the probabilistic model of the dynamical system: (1) A *measurement model* $p(z_t|x_t)$, which describes the probability of measuring $z_t$ when the system is in state $x_t$. (2) A *control model* $p(x_t|u_t, x_{t-1})$, which characterizes the effect of controls $u_t$ on the system state by specifying the probability that the system is in state $x_t$ after executing control $u_t$ in state $x_{t-1}$. (3) An *initial state distribution* $p(x_0)$, which specifies the user's knowledge about the initial system state. See [2, 5] for examples of such models in practical applications.

Eqn. 1 is easily derived under the common assumption that the system is Markov:

$$
\begin{aligned}
p(x_t|z^t, u^t) \quad &\overset{\text{Bayes}}{=} \quad \eta_t \, p(z_t|x_t, z^{t-1}, u^t) \, p(x_t|z^{t-1}, u^t) \\
&\overset{\text{Markov}}{=} \quad \eta_t \, p(z_t|x_t) \, p(x_t|z^{t-1}, u^t) \\
&= \quad \eta_t \, p(z_t|x_t) \int p(x_t|z^{t-1}, u^t, x_{t-1}) \, p(x_{t-1}|z^{t-1}, u^t) \, dx_{t-1} \\
&\overset{\text{Markov}}{=} \quad \eta_t \, p(z_t|x_t) \int p(x_t|u_t, x_{t-1}) \, p(x_{t-1}|z^{t-1}, u^{t-1}) \, dx_{t-1} \quad (2)
\end{aligned}
$$

Notice that this filter, in the general form stated here, is commonly known as a Bayes filter. Approximations to Bayes filters includes the Kalman filter, the hidden Markov model, binary filters, and of course particle filters. In many applications, the key concern in implementing this probabilistic filter is the continuous nature of the states $x$, controls $u$, and measurements $z$. Even in discrete applications, the state space is often too large to compute the entire posterior in reasonable time.

The particle filter addresses these concerns by approximating the posterior using sets of state samples (particles):

$$X_t = \{x_t^{[i]}\}_{i=1,\ldots,m} \qquad (3)$$

The set $X_t$ consists of $m$ particles $x_t^{[i]}$, for some large number $m$ (e.g, $m = 1,000$). Together, these particles approximates the posterior $p(x_t|z^t, u^t)$. $X_t$ is calculated recursively. Initially, at time $t = 0$, the particles $x_0^{[i]}$ are generated from the initial state distribution $p(x_0)$. The $t$-th particle set $X_t$ is then calculated recursively from $X_{t-1}$ as follows:

```
1           set $X_t = X_t^{\mathrm{aux}} = \emptyset$
2           for $j = 1$ to $m$ do
3               pick the $j$-th sample $x_{t-1}^{[j]} \in X_{t-1}$
4               draw $x_t^{[j]} \sim p(x_t|u_t, x_{t-1}^{[j]})$
5               set $w_t^{[j]} = p(z_t|x_t^{[j]})$
6               add $\langle x_t^{[j]}, w_t^{[j]} \rangle$ to $X_t^{\mathrm{aux}}$
7           endfor
8           for $i = 1$ to $m$ do
9               draw $x_t^{[i]}$ from $X_t^{\mathrm{aux}}$ with probability proportional to $w_t^{[i]}$
10              add $x_t^{[i]}$ to $X_t$
11          endfor
```

Lines 2 through 7 generates a new set of particles that incorporates the control $u_t$. Lines 8 through 11 apply a technique known as *importance-weighted resampling* [11] to account for the measurement $z_t$. It is a well-known fact that (for large $m$) the resulting weighted particles are asymptotically distributed according to the desired posterior [12] $p(x_t|z^t, u^t)$

In recent years, researchers have actively developed various extensions of the basic particle filter, capable of coping with degenerate situations that are often relevant in practice [3, 7, 8]. The common aim of this rich body of literature, however, is to generate samples from the posterior $p(x_t|z^t, u^t)$. If different controls at different states infer drastically different costs, generating samples according to the posterior runs the risk of not capturing important events that warrant action. Overcoming this deficiency is the very aim of this paper.

## 3   Risk Sensitive Particle Filters

This section describes a modified particle filter that is sensitive to the risk arising from the approximate nature of the particle representation. To arrive at a notion of risk, our approach requires a cost function

$$C(x, u) \in \Re \tag{4}$$

This function assigns real-valued costs to states and control. From a decision theoretic point of view, the goal of risk sensitive sampling is to generate particles that minimize the cumulative increase in cost due to the particle approximation. To translate this into a practical algorithm, we extend the basic paradigm in two ways. First, we modify the basic particle filters so that particles are generated in a risk-sensitive way, where the risk is a function of $C$. Second, an appropriate risk function is defined that approximates the cumulative expected costs relative to tracking individual states. This risk function is calculated using value iteration.

### 3.1   Risk-Sensitive Sampling

Risk-sensitive sampling generates particles factoring in a *risk function*, $r(x)$. Formally, all we have to ask of a risk function $r$ is that it be positive and finite almost everywhere. Not all risk functions will be equally useful, however, so deriving the "right" risk function is important. Decision theory gives us a framework for deciding what the "right" action is in any given state. By considering approximation errors due to monte carlo sampling in decision theory and making a sequence of rough approximations, we can arrive at the choice of $r(x)$, which is discussed further below. The full derivation is omitted for lack of space. For now, let us simply assume are given a suitable risk function.

Risk sensitive particle filters generate samples that are distributed according to

$$\gamma_t \; r(x_t) \; p(x_t|z^t, u^t) \tag{5}$$

Here $\gamma_t = [\int r(x)p(x|z^t, u^t)dx]^{-1}$ is a normalization constant that ensures that the term in (5) is indeed a probability distribution. Thus, the probability that a state sample $x_t^{[i]}$ is part of $X_t$ is not only a function of its posterior probability, but also of the risk $r(x_t^{[i]})$ associated with that sample.

Sampling from (5) is easily achieved by the following two modifications of the basic particle filter algorithm. First, the initial set of particles $x_0^{[i]}$ is generated from the distribution

$$\gamma_0 \; r(x_0) \; p(x_0) \tag{6}$$

Second, Line 5 of the particle filter algorithm is replaced by the following assignment:

$$\text{set } \; w_t^{[j]} = r(x_t^{[j]}) \; r(x_{t-1}^{[j]})^{-1} \; p(z_t|x_t^{[j]}) \tag{7}$$

We conjecture that this simple modification results in a particle filter with samples distributed according to $\gamma_t r(x_t)p(x_t|z^t, u^t)$. Our conjecture is obviously true for the base case $t=0$, since the risk function $r$ was explicitly incorporated in the construction of $X_0$ (see eqn. 6). By induction, let us assume that the particles in $X_{t-1}$ are distributed according to $\gamma_{t-1} \; r(x_{t-1}) \; p(x_{t-1}|z^{t-1}, u^{t-1})$. Then Line 3 of the modified algorithm generates $x_{t-1}^{[j]} \sim \gamma_{t-1} \; r(x_{t-1}) \; p(x_{t-1}|z^{t-1}, u^{t-1})$. Line 4 gives us $x_t^{[j]} \sim \gamma_{t-1} \; r(x_{t-1}) \; p(x_t|u_t, x_{t-1}) \; p(x_{t-1}|z^{t-1}, u^{t-1})$. Samples generated in Line 9 are distributed according to

$$w_t^{[j]}\gamma_{t-1} \; r(x_{t-1}) \; p(x_t|u_t, x_{t-1}) \; p(x_{t-1}|z^{t-1}, u^{t-1}) \tag{8}$$

Substituting in the modified weight (eqn. 7) we find the final sample distribution:

$$
\begin{aligned}
& r(x_t) \; r(x_{t-1})^{-1} \; p(z_t|x_t) \; \gamma_{t-1} \; r(x_{t-1}) \; p(x_t|u_t, x_{t-1}) \; p(x_{t-1}|z^{t-1}, u^{t-1}) \\
& = \quad \gamma_{t-1} \; r(x_t) \; p(z_t|x_t) \; p(x_t|u_t, x_{t-1}) \; p(x_{t-1}|z^{t-1}, u^{t-1})
\end{aligned} \tag{9}
$$

This term is, up to the normalization constant $\gamma_t \eta_t \gamma_{t-1}^{-1}$, equivalent to the desired distribution (5) (see also eqn. 1), which proves our conjecture. Thus, the risk sensitive particle filter successfully generates samples from a distribution that factors in the risk $r$.

### 3.2 The Risk Function

The remaining question is: What is an appropriate risk function $r$? How important is it to track a state $x$? Our approach rests on the assumption that there are two possible situations, one in which the state is tracked well, and one in which the state is tracked poorly. In the first situation, we assume that any controller will basically chose the right control, whereas in the second situation, it is reasonable to assume that controls are selected anywhere between random and in the worst possible way. To complete this model, we assume that with small probability, the state estimator might move from "well-tracked" to "lost track" and vice versa.

These assumptions are sufficient to formulate an MDP that models the effect of tracking accuracy on the expected costs. The MDP is defined over an augmented state space $\langle x, c \rangle$ (see also [10]), where $c \in \{0, 1\}$ is a binary state variable that models the event that the estimator tracks the state with sufficient ($c_t=1$) or insufficient ($c_t=0$) accuracy. The various probabilities of the MDP are easily obtained from the known probability distributions via

the natural assumption that the variable $c$ is conditionally independent of the system state $x$:

$$
\begin{aligned}
p(\langle x_t, c_t \rangle | u_t, \langle x_{t-1}, c_{t-1} \rangle) &= p(x_t | u_t, x_{t-1})\, p(c_t | c_{t-1}) \\
p(z_t | \langle x_t, c_t \rangle) &= p(z_t | x_t) \\
p(\langle x_0, c_0 \rangle) &= p(x_0)\, p(c_0) \\
C(\langle x_t, c_t \rangle, u_t) &= C(x_t, u_t)
\end{aligned}
\tag{10}
$$

The expressions on the left hand side define all necessary components of the augmented model. The only unspecified terms on the right hand side are the initial tracking probability $p(c_0)$ and the transition probabilities for the state estimator $p(c_t | c_{t-1})$. The former must be set in accordance to the initial knowledge state (e.g., 1 if the initial system state is known, 0 if it is unknown). For the latter, we adopt a model where with high likelihood the tracking state is retained ($p(c_t{=}c_{t-1}) = 0.95$) and with low likelihood it changes ($p(c_t{\neq}c_{t-1}) = 0.05$).

The MDP is solved via value iteration. To model the effect of poor tracking on the control policy, our approach uses the following value iteration rule (stated here without discounting for simplicity), in which $V$ denotes the value function, and $Q$ is an auxiliary variable:

$$
V(\langle x, c \rangle) = \begin{cases}
\min\limits_{u} Q(\langle x, c \rangle, u) & \text{if } c{=}1 \\[2mm]
\beta\, [\max\limits_{u} Q(\langle x, c \rangle, u)] \; + \; (1{-}\beta)\, [\int Q(\langle x, c \rangle, u)\, du] & \text{if } c{=}0
\end{cases}
$$

$$
Q(\langle x, c \rangle, u) = C(x, u) + \sum_{c'=0}^{1} \int V(\langle x', c' \rangle)\, p(c'|c)\, p(x'|u, x)\, dx'
\tag{11}
$$

This value iteration rule considers two cases: When $c{=}1$, i.e., the state is estimated sufficiently accurately, it is assumed that the controller acts by minimizing costs. If $c{=}0$, however, the controller adopts a mixture of picking the *worst* possible control $u$, and a random control. These two options are traded off by the gain factor $\beta$, which controls the "pessimism" of the approach. $\beta{=}1$ suggests that poor state estimation leads to the worst possible control. $\beta{=}0$ is more optimistic, in that control is assumed to be random. Our experiments have yielded somewhat indifferent results relative to the choice of $\beta$, and we use $\beta{=}0.5$ for all experiments reported here.

Finally, the risk $r$ is defined as the difference between the value function that arises from accurate versus inaccurate state estimation:

$$
r(x) = V(x, c = 0) - V(x, c = 1)
\tag{12}
$$

Under mild assumptions, $r(x)$ can be shown to be strictly positive.

## 4   Experimental Results

We have applied our approach to two complimentary real-world robotic domains: robot localization, and mobile robot diagnostics. Both yield superior results using our new risk sensitive approach when compared to the standard particle filter.

### 4.1   Mobile Robot Localization

Our first evaluation domain involves the problem of localizing a mobile robot from sensor data [2]. In our experiments, we focused on the most difficult of all localization problems:

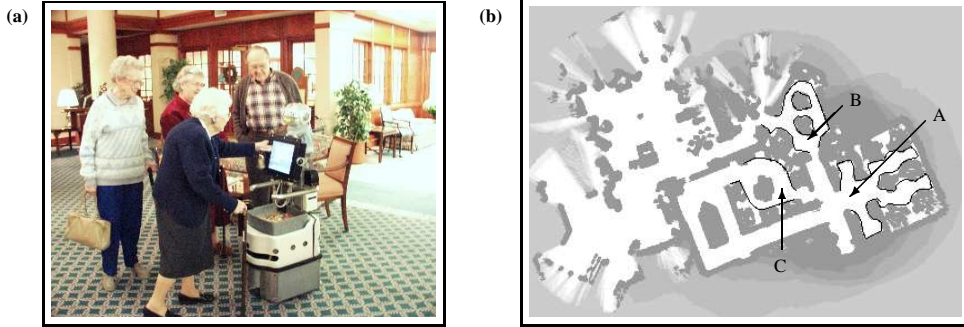

**Figure 1**: (a) Robot Pearl, as it interacts with elderly people at an assisted living facility in Oakmont, PA. (b) Occupancy grid map. Shown here are also three testing locations labeled A, B, and C, and regions of high costs (black contours).

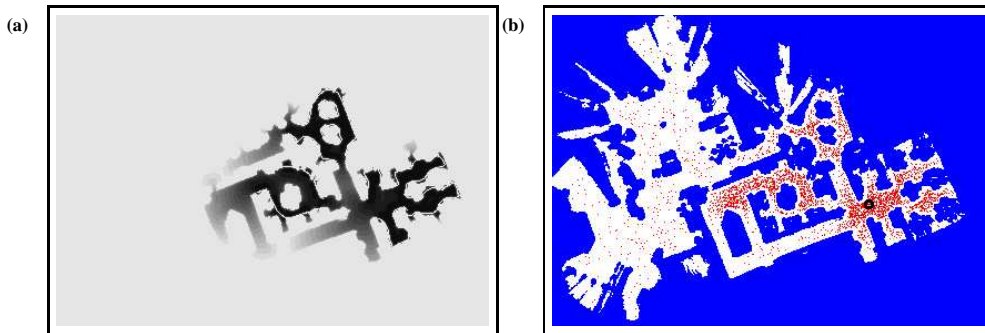

**Figure 2**: (a) Risk function $r$: the darker a location, the higher the risk. This function, which is used in the proposal distribution, is derived from the immediate risk function shown in Figure 1b. (b) Sample of a uniform distribution, taking into consideration the risk function.

|  | standard filter | risk sensitive filter |
|---|---|---|
| steps to re-localize when ported to A | $120 \pm 13.7$ | $89.3 \pm 12.3$ |
| steps to re-localize when ported to B | $301 \pm 35.2$ | $203 \pm 37.6$ |
| steps to re-localize when ported to C | $63.2 \pm 6.2$ | $53.2 \pm 7.7$ |
| number of violations after global kidnapping | $96.1 \pm 14.1$ | $57.4 \pm 10.3$ |

**Table 1**: Localization results for the *kidnapped robot problem*, which emulates a total localization failure. Our new approach requires consistently fewer steps for re-localization in high-cost areas, and therefore incurs less cost.

The kidnapped robot problem [4]. Here a well-localized robot is "tele-ported" to some unknown location and has to recover from this event. This problem plays an important role in evaluating the robustness of a localization algorithm. Figure 1a shows the robot Pearl, which has recently been deployed in an assisted living facility as an assistant to the elderly and cognitively frail. Our study is motivated by the fact that some of the robot's operational area is a densely cluttered dining room, where the robot is not allowed to cross certain boundaries due to the danger of physically harming people. These boundaries are illustrated by the black contours shown in Figure 1b, which also depicts an occupancy grid map of the facility. Beyond the boundaries, the robot's sensor are somewhat insufficient to avoid collisions, since they can only sense obstacles at one specific height (34 cm).

Figure 2a shows the risk function $r$, projected into 2D. The darker a location, the higher the risk. A sample set drawn from this risk function is shown in Figure 2b. This sample set represents a uniform posterior. Since risk sensitive particle filters incorporate the risk

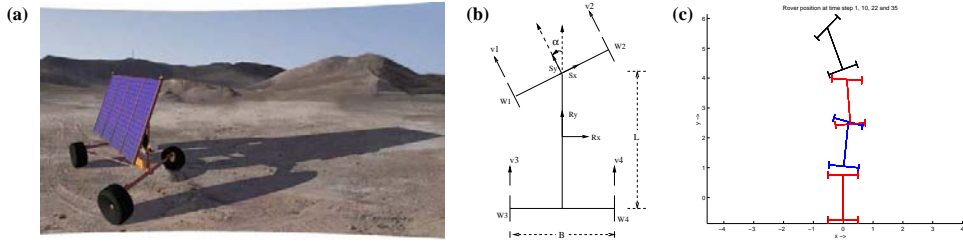

**Figure 3**: (a) The Hyperion rover, a mobile robot being developed at CMU. (b) Kinematic model. (c) Rover position at time step 1, 10, 22 and 35.

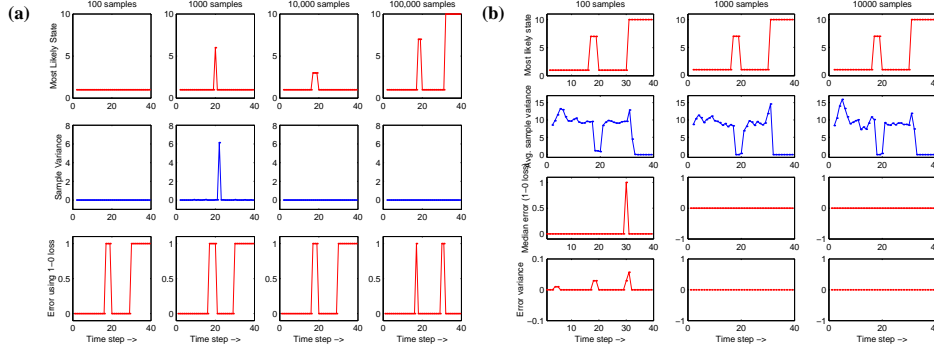

**Figure 4**: Tracking curves obtained with (a) plain particle filters, and (b) our new risk sensitive filter. The bottom curves show the error, which is much smaller for our new approach.

function into the sampling process, however, the density of samples is proportional to the risk function $r$.

Numerical results are summarized in Table 1, using data collected in the facility at dinner time. We ran two types of experiments: First, we kidnapped the robot to any of the locations marked A, B, and C in Figure 1, and measured the number of sensor readings required to recover from this global failure. All three locations are within the high-risk area so the recovery time is significantly shorter than with plain particle filters. Second, we measured the number of times a simple-minded planner that always looks at the most likely pose would violate the safety constraint. Here we find that our approach is almost twice as safe as the conventional particle filter, at virtually the same computational expense. All experiments were repeated 20 times, and rely on real-world data and operating conditions.

## 4.2 Mobile Robot Diagnosis

In some domains, particle filters simply cannot be applied in real time because of a large number of high loss and low probability events. One example is the fault detection domain illustrated in Figure 3. Our evaluation involves a data set where a rover is driven with a variety of different control inputs in the normal operation mode. At the $17th$ time step, wheel #3 becomes stuck and locked against a rock. The wheel is then driven in the backward direction, fixing the problem. The rover returns to the normal operation mode and continues to operate normally until the gear on wheel #4 breaks at the $30th$ time step. This fault is not recoverable and the controller just alters its input based on this state. Notice that both failures lead to very similar sensor measurement, despite the fact that they are caused by quite different events.

Tracking results in Figure 4 show that our approach yields superior results to the standard particle filter. Even though failures are very unlikely, our approach successfully identifies them due to the high risk associated with such a failure while the plain particle filter essentially fails to do so. The estimation error is shown in the bottom row of Figure 4, which is practically zero for our approach when 1,000 or more samples are used. Vanialle particle filters exhibit non-zero error even with 100,000 samples. However, it is important to notice that these results were obtained using simulated data and a hand-tuned loss function approach.

## 5 Discussion

We have proposed a particle filter algorithm that considers a cost model when generating samples. The key idea is that particles are generated in proportion to their posterior likelihood *and* to the risk that arises relative to a control goal. An MDP algorithm was developed that computes the risk function as a differential cumulative cost. Experimental results in two robotic domains show the superior performance of our new approach.

An alternative approach for solving the problem addressed in this paper would be to analyze the estimation process as a partially observable Markov decision process (POMDP) [6]. Bounds on the performance loss due to the approximate nature of particle filters can be found in [9]. Pursuing the problem of risk-sensitive particle generation within the POMDP framework might be a promising future line of research.

### Acknowledgment

The authors thank Dieter Fox and Wolfram Burgard, who generously provided some the localization software on which this research is built. Financial support by DARPA (TMR, MARS, CoABS and MICA programs) and NSF (ITR, Robotics, and CAREER programs) is gratefully acknowledged.

## References

[1] X. Boyen and D. Koller. Tractable inference for complex stochastic processes. In *Proc. UAI-98*.

[2] F. Dellaert, D. Fox, W. Burgard, and S. Thrun. Monte carlo localization for mobile robots. In *Proc. ICRA-99*.

[3] A. Doucet, J.F.G. de Freitas, and N.J. Gordon, editors. *Sequential Monte Carlo Methods In Practice*. Springer, 2001.

[4] S. Engelson. *Passive Map Learning and Visual Place Recognition*. PhD thesis, Computer Science Department, Yale University, 1994.

[5] M. Isard and A. Blake. CONDENSATION: conditional density propagation for visual tracking. *International Journal of Computer Vision*, 29(1):5–28, 1998.

[6] L.P. Kaelbling, M.L. Littman, and A.R. Cassandra. Planning and acting in partially observable stochastic domains. *Artificial Intelligence*, 101(1-2):99–134, 1998.

[7] J. Liu and R. Chen. Sequential monte carlo methods for dynamic systems. *Journal of the American Statistical Association*, 93:1032–1044, 1998.

[8] M. Pitt and N. Shephard. Filtering via simulation: auxiliary particle filter. *Journal of the American Statistical Association*, 94:590–599, 1999.

[9] P. Poupart, L.E. Ortiz, and C. Boutilier. Value-directed sampling methods for monitoring POMDPs. In *Proc. UAI-2001*.

[10] N. Roy and S. Thrun. Coastal navigation with mobile robot. In *Proc. NIPS-99*.

[11] D.B. Rubin. Using the SIR algorithm to simulate posterior distributions. In *Bayesian Statistics 3*. Oxford Univ. Press, 1988.

[12] M.A. Tanner. *Tools for Statistical Inference*. Springer, 1996.
